# The Recurrent Cascade-Correlation Architecture

**Scott E. Fahlman**
School of Computer Science
Carnegie Mellon University
Pittsburgh, PA 15213

## Abstract

Recurrent Cascade-Correlation (RCC) is a recurrent version of the Cascade-Correlation learning architecture of Fahlman and Lebiere [Fahlman, 1990]. RCC can learn from examples to map a sequence of inputs into a desired sequence of outputs. New hidden units with recurrent connections are added to the network as needed during training. In effect, the network builds up a finite-state machine tailored specifically for the current problem. RCC retains the advantages of Cascade-Correlation: fast learning, good generalization, automatic construction of a near-minimal multi-layered network, and incremental training.

## 1 THE ARCHITECTURE

Cascade-Correlation [Fahlman, 1990] is a supervised learning architecture that builds a near-minimal multi-layer network topology in the course of training. Initially the network contains only inputs, output units, and the connections between them. This single layer of connections is trained (using the Quickprop algorithm [Fahlman, 1988]) to minimize the error. When no further improvement is seen in the level of error, the network's performance is evaluated. If the error is small enough, we stop. Otherwise we add a new hidden unit to the network in an attempt to reduce the residual error.

To create a new hidden unit, we begin with a pool of *candidate units*, each of which receives weighted connections from the network's inputs and from any hidden units already present in the net. The outputs of these candidate units are not yet connected into the active network. Multiple passes through the training set are run, and each candidate unit adjusts its incoming weights to maximize the correlation between its output and the residual error in the active net. When the correlation scores stop improving, we choose the best candidate, freeze its incoming weights, and add it to the network. This process is called "tenure." After tenure,

a unit becomes a permanent new feature detector in the net. We then re-train all the weights going to the output units, including those from the new hidden unit. This process of adding a new hidden unit and re-training the output layer is repeated until the error is negligible or we give up. Since the new hidden unit receives connections from the old ones, each hidden unit effectively adds a new layer to the net.

Cascade-correlation eliminates the need for the user to guess in advance the network's size, depth, and topology. A reasonably small (though not minimal) network is built automatically. Because a hidden-unit feature detector, once built, is never altered or cannibalized, the network can be trained incrementally. A large data set can be broken up into smaller "lessons," and feature-building will be cumulative. Cascade-Correlation learns much faster than backprop for several reasons: First only a single layer of weights is being trained at any given time. There is never any need to propagate error information backwards through the connections, and we avoid the dramatic slowdown that is typical when training backprop nets with many layers. Second, this is a "greedy" algorithm: each new unit grabs as much of the remaining error as it can. In a standard backprop net, the all the hidden units are changing at once, competing for the various jobs that must be done—a slow and sometimes unreliable process.

Cascade-correlation, like back-propagation and other feed-forward architectures, has no short-term memory in the network. The outputs at any given time are a function only of the current inputs and the network's weights. Of course, many real-world tasks require the recognition of a sequence of inputs and, in some cases, the corresponding production of a sequence of outputs. A number of recurrent architectures have been proposed in response to this need. Perhaps the most widely used, at present, is the Elman model [Elman, 1988], which assumes that the network operates in discrete time-steps. The outputs of the network's hidden units at time $t$ are fed back for use as additional network inputs at time-step $t+1$. These additional inputs can be thought of as state-variables whose contents and interpretation are determined by the evolving weights of the network. In effect, the network is free to choose its own representation of past history in the course of learning.

Recurrent Cascade-Correlation (RCC) is an architecture that adds Elman-style recurrent operation to the Cascade-Correlation architecture. However, some changes were needed in order to make the two models fit together. In the original Elman architecture there is total connectivity between the state variables (previous outputs of hidden units) and the hidden unit layer. In Cascade-Correlation, new hidden units are added one by one, and are frozen once they are added to the network. It would violate this concept to insert the outputs from new hidden units back into existing hidden units as new inputs. On the other hand, the network must be able to form recurrent loops if it is to retain state for an indefinite time.

The solution we have adopted in RCC is to augment each candidate unit with a single weighted self-recurrent input that feeds back that unit's own output on the previous time-step. That self-recurrent link is trained along with the unit's other input weights to maximize the correlation of the candidate with the residual error. If the recurrent link adopts a strongly positive value, the unit will function as a flip-flop, retaining its previous state unless the other inputs force it to change; if the recurrent link adopts a negative value, the unit will tend to oscillate between positive and negative outputs on each time-step unless the other inputs hold it in place; if the recurrent weight is near zero, then the unit will act as a gate of some kind. When a candidate unit is added to the active network as a new hidden unit, the self-recurrent weight is frozen, along with all the other weights. Each new hidden unit is in effect a single state variable in a finite-state machine that is built specifically for the

task at hand. In this use of self-recurrent connections only, the RCC model resembles the "Focused Back-Propagation" algorithm of Mozer[Mozer, 1988].

The output, $V(t)$, of each self-recurrent unit is computed as follows:

$$V(t) = \sigma \left( \sum_i I_i(t)\, w_i \ + \ V(t-1)\, w_s \right)$$

where $\sigma$ is some non-linear squashing function applied to the weighted sum of inputs $I$ plus the self-weight, $w_s$, times the previous output. In the studies described here, $\sigma$ is always the hyperbolic tangent or "symmetric sigmoid" function, with a range from -1 to +1. During the candidate training phase, we adjust the weights $w_i$ and $w_s$ for each unit so as to maximize its correlation score. This requires computing the derivative of $V(t)$ with respect to these weights:

$$\partial V(t)/\partial w_i = \sigma'(t) \ \left( I_i(t) + w_s \ \partial V(t-1)/\partial w_i \right)$$

$$\partial V(t)/\partial w_s = \sigma'(t) \ \left( V(t-1) + w_s \ \partial V(t-1)/\partial w_s \right)$$

The rightmost term reflects the influence of the weight in question on the unit's previous state. Since we computed $\partial V(t-1)/\partial w$ on the previous time-step, we can just save this value and use it in the current step. So the recurrent version of the learning algorithm requires us to store a single additional number for each candidate weight, plus $V(t-1)$ for each unit. At $t = 0$ we assume that the unit's previous value and previous derivatives are all zero.

As an aside, the usual formulation for Elman networks treats the hidden units' previous values as *independent* inputs, ignoring the dependence of these previous values on the weights being adjusted. In effect, the rightmost terms in the above equations are being dropped, though they are not negligible in general. This rough approximation apparently causes little trouble in practice, but it might explain the instability that some researchers have reported when Elman nets are run with aggressive second-order learning procedures such as quickprop. The Mozer algorithm does take these extra terms into account.

## 2   EMPIRICAL RESULTS: FINITE-STATE GRAMMAR

Figure 1a shows the state-transition diagram for a simple finite-state grammar, called the Reber grammar, that has been used by other researchers to investigate learning and generalization in recurrent neural networks. To generate a "legal" string of tokens from this grammar, we begin at the left side of the graph and move from state to state, following the directed edges. When an edge is traversed, the associated letter is added to the string. Where two paths leave a single node, we choose one at random with equal probability. The resulting string always begins with a "B" and ends with an "E". Because there are loops in the graph, there is no bound on the length of the strings; the average length about eight letters. An example of a legal string would be "BTSSXXVPSE".

Cleeremans, Servan-Schreiber, and McClelland [Cleeremans, 1989] showed that an Elman network can learn this grammar if it is shown many different strings produced by the

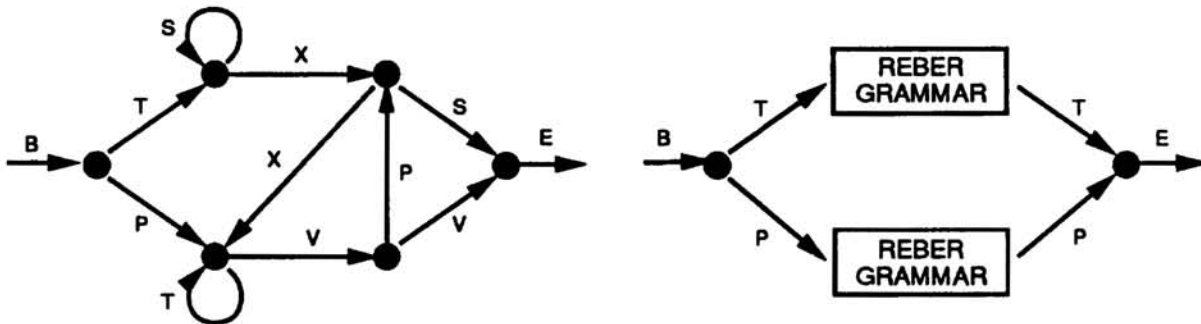

Figure 1: State transition diagram for the Reber grammar (left) and for the embedded Reber grammar (right).

grammar. The internal state of the network is zeroed at the start of each string. The letters in the string are then presented sequentially at the inputs of the network, with a separate input connection for each of the seven letters. The network is trained to predict the next character in the string by turning on one of the seven outputs. The output is compared to the true successor and network attempts to minimize the resulting errors.

When there are two legal successors from a given state, the network will never be able to do a perfect job of prediction. During training, the net will see contradictory examples, sometimes with one successor and sometimes the other. In such cases, the net will eventually learn to partially activate both legal outputs. During testing, a prediction is considered correct if the two desired outputs are the two with the largest values.

This task requires generalization in the presence of considerable noise. The rules defining the grammar are never presented—only examples of the grammar's output. Note that if the network can perform the prediction task perfectly, it can also be used to determine whether a string is a legal output of the grammar. Note also that the successor letter(s) cannot be determined from the current input alone; some memory of of the network's state or past inputs is essential.

Cleeremans *et al.* report that a fixed-topology Elman net with three hidden units can learn this task after 60,000 distinct training strings have been presented, each used only once. A larger network with 15 hidden units required only 20,000 training strings. These were the best results obtained, not averages over a number of runs.

RCC was given the same problem, but using a fixed set of 128 training strings, presented repeatedly. (Smaller string-sets had too many statistical irregularities for reliable training.) Ten trials were run using different training sets. In nine cases, RCC achieved perfect performance after building two hidden units; in the tenth, three hidden units were built. Average training time was 195.5 epochs, or about 25,000 string presentations. (An *epoch* is defined as a single pass through a fixed training set.) In every case, the trained network achieved a perfect score on a set of 128 new strings not used in training. This study used a pool of 8 candidate units.

Cleeremans *et al.* also explored the "embedded Reber grammar" shown in figure 1b. Each

of the boxes in the figure is a transition graph identical to the original Reber grammar. In this much harder task, the network must learn to predict the final T or P correctly. To accomplish this, the network must note the initial T or P and must retain this information while processing an "embedded clause" of arbitrary length. It is difficult to discover this rule from example strings, since the embedded clauses may also contain many T's and P's, but only the initial T or P correlates reliably with the final prediction. The "signal to noise ratio" in this problem is very poor.

The standard Elman net was unable to learn this task, even with 15 hidden units and 250,000 training strings. However, the task could be learned partially (correct prediction in about 70% of test strings) if the two copies of the embedded grammar were differentiated by giving them slightly different transition probabilities.

RCC was run six times on the more difficult symmetrical form of this problem. A candidate pool of 8 units was used. Each trial used a different set of 256 training strings and the resulting network was tested on a separate set of 256 strings. As shown in the table below, perfect performance was achieved in about half the trial runs, requiring 7-9 hidden units and and average of 713 epochs (182K string-presentations). Two of the remaining networks perform at the 99+% level, and one got stuck. (Trial 6 is a successful second run on the same test set used in trial 5.)

| Trial | Hidden Units | Epochs Needed | Train Set Errors | Test Set Errors |
|:---:|:---:|:---:|:---:|:---:|
| 1 | 9 | 831 | 0 | 0 |
| 2 | 7 | 602 | 0 | 0 |
| 3 | 15 | 1256 | 0 | 2 |
| 4 | 11 | 910 | 0 | 1 |
| 5 | 13 | 1063 | 11 | 16 |
| 6 | 9 | 707 | 0 | 0 |

Smith and Zipser[Smith, 1989] have studied the same grammar-learning tasks using the time-continuous "Real-Time Recurrent Learning" (or "RTRL") architecture developed by Williams and Zipser[Williams, 1989]. They report that a network with seven visible (combined input-output) units, two hidden units, and full inter-unit connectivity is able to learn the simple Reber grammar task after presentation of 19,000 to 63,000 distinct training strings. On the more difficult embedded grammar task, Smith and Zipser report that RTRL learned the task perfectly in some (unspecified) fraction of attempts. Successful runs ranged from 3 hidden units (173K distinct training strings) to 12 hidden units (25K strings). RTRL is able to deal with discrete or time-continuous problems, while RCC deals only in discrete events. On the other hand, RTRL requires more computation than RCC in processing each training example, and RTRL scales up poorly as network size increases.

## 3   EMPIRICAL RESULTS: LEARNING MORSE CODE

Another series of experiments tested the ability of an RCC network to learn the Morse code patterns for the 26 English letters. While this task requires no generalization, it does demonstrate the ability of this architecture to recognize a long, rather complex set of patterns. It also provides an opportunity to demonstrate RCC's ability to learn a new task in small increments. This study assumes that the dots and dashes arrive at precise times; it does not address the problem of variable timing.

The network has one input and 27 outputs: one for each letter and a "strobe" output signalling that a complete letter has been recognized. A dot is represented as a logical one (positive input) followed by a logical zero (negative); a dash is two ones followed by a zero. A second consecutive zero marks the end of the letter. When the second zero is seen the network must raise the strobe output and one of the other 26; at all other times, the outputs are zero. For example, the "...-" pattern for the letter V would be encoded as the input sequence "1010101100". The letter patterns vary considerably in length, from 3 to 12 time-steps, with an average of 8. During training, the network's state is zeroed at the start of each new letter; once the network is trained, the strobe output could be used to reset the network.

In one series of trials, the training set included the codes for all 26 letters at once (226 time-steps in all). In ten trials, the network learned the task perfectly in every case, building an average of 10.5 hidden units and requiring an average of 1321 passes through the entire training set. Note that the system does not require a distinct hidden unit for each letter or for each time-slice in the longest sequence.

In a second experiment, we divided the training into a series of short "lessons" of increasing difficulty. The network was first trained to produce the strobe output and to recognize the two shortest letters, E and T. This task was learned perfectly, usually with the creation of 2 hidden units. We then set aside the "ET" set and trained successively on the following sets: "AIN", "DGHKRUW", "BFLOV", and "CJPQXYZ". As a rule, each of these lessons adds one or two new hidden units, building upon those already present. Finally we train on all 26 characters at once, which generally adds 2-3 more units to the existing set.

In ten trials, the incremental version learned the task perfectly every time, requiring an average total of 1427 epochs and 9.6 hidden units—slightly fewer than the number of units added in block training. While the epoch count is slightly greater than in the block-training experiment, most of these epochs are run on very small training sets. The incremental training required only about half as much total runtime as the block training. For learning of even more complex temporal sequences, incremental training of this kind may prove essential.

Our approach to incremental training was inspired to some degree by the work reported in [Waibel, 1989] in which small network modules were trained separately, frozen, and then combined into a composite network with the addition of some "glue" units. However, in RCC only the partitioning of the training set is chosen by the user; the network itself builds the appropriate internal structure, and new units are able to build upon hidden units created during some earlier lesson.

## 4  CONCLUSIONS

RCC sequential processing to Cascade-Correlation, while retaining the advantages of the original version: fast learning, good generalization, automatic choice of network topology, ability to create complex high-order feature detectors, and incremental learning. The grammar-learning experiments suggest that RCC is more powerful than standard Elman networks in learning to recognize subtle patterns in sequential data. The RTRL scheme of Williams and Zipser may be equally powerful, but RTRL is more complex and does not scale up well when larger networks are needed.

On the negative side, RCC deals in discrete time-steps and not in continuous time. An

interesting direction for future research is to explore the use of an RCC-like structure with units whose memory of past state is time-continuous rather than discrete.

## Acknowledgments

I would like to thank Paul Gleichauf, Dave Touretzky, and Alex Waibel for their help and useful suggestions. This research was sponsored in part by the National Science Foundation (Contract EET-8716324) and the Defense Advanced Research Projects Agency (Contract F33615-90-C-1465).

# References

[Cleeremans, 1989] Cleeremans, A., D. Servan-Schreiber, and J. L. McClelland (1989) "Finite-State Automata and Simple Recurrent Networks" in *Neural Computation 1*, 372-381.

[Elman, 1988] Elman, J. L. (1988) "Finding Structure in Time," CRL Tech Report 8801, Univ. of California at San Diego, Center for Research in Language.

[Fahlman, 1988] Fahlman, S. E. (1988) "Faster-Learning Variations on Back-Propagation: An Empirical Study" in *Proceedings of the 1988 Connectionist Models Summer School*, Morgan Kaufmann.

[Fahlman, 1990] Fahlman, S. E. and C. Lebiere (1988) "The Cascade-Correlation Learning Architecture" in D. S. Touretzky (ed.), *Advances in Neural Information Processing Systems 2*, Morgan Kaufmann.

[Mozer, 1988] Mozer, M. C. (1988) "A Focused Back-Propagation Algorithm for Temporal Pattern Recognition," Tech Report CRG-TR-88-3, Univ. of Toronto, Dept. of Psychology and Computer Science.

[Smith, 1989] Smith, A. W. and D. Zipser (1989) "Learning Sequential Structure with the Real-Time Recurrent Learning Algorithm" in *International Journal of Neural Systems*, Vol. 1, No. 2, 125-131.

[Waibel, 1989] Waibel, A. (1989) "Consonant Recognition by Modular Construction of Large Phonemic Time-Delay Neural Networks" in D. S. Touretzky (ed.), *Advances in Neural Information Processing Systems 1*, Morgan Kaufmann.

[Williams, 1989] Williams, R. J. and D. Zipser (1989) "A learning algorithm for continually running fully recurrent neural networks," Neural Computation 1, 270-280.


